# Learning with Preknowledge: Clustering with Point and Graph Matching Distance Measures

**Steven Gold**[1], **Anand Rangarajan**[1] **and Eric Mjolsness**[2]
Department of Computer Science
Yale University
New Haven, CT 06520-8285

## Abstract

Prior constraints are imposed upon a learning problem in the form of distance measures. Prototypical 2-D point sets and graphs are learned by clustering with point matching and graph matching distance measures. The point matching distance measure is approx. invariant under affine transformations - translation, rotation, scale and shear - and permutations. It operates between noisy images with missing and spurious points. The graph matching distance measure operates on weighted graphs and is invariant under permutations. Learning is formulated as an optimization problem. Large objectives so formulated ($\sim$ million variables) are efficiently minimized using a combination of optimization techniques - algebraic transformations, iterative projective scaling, clocked objectives, and deterministic annealing.

## 1 Introduction

While few biologists today would subscribe to Locke's description of the nascent mind as a tabula rasa, the nature of the inherent constraints - Kant's preknowl-

edge - that helps organize our perceptions remains much in doubt. Recently, the importance of such preknowledge for learning has been convincingly argued from a statistical framework [Geman et al., 1992]. Researchers have proposed that our brains may incorporate preknowledge in the form of distance measures [Shepard, 1989]. The neural network community has begun to explore this idea via tangent distance [Simard et al., 1993], model learning [Williams et al., 1993] and point matching distances [Gold et al., 1994]. However, only the point matching distances have been invariant under permutations. Here we extend that work by enhancing both the scope and function of those distance measures, significantly expanding the problem domains where learning may take place.

We learn objects consisting of noisy 2-D point-sets or noisy weighted graphs by clustering with point matching and graph matching distance measures. The point matching measure is approx. invariant under permutations and affine transformations (separately decomposed into translation, rotation, scale and shear) and operates on point-sets with missing or spurious points. The graph matching measure is invariant under permutations. These distance measures and others like them may be constructed using Bayesian inference on a probabilistic model of the visual domain. Such models introduce a carefully designed bias into our learning, which reduces its generality outside the problem domain but increases its ability to generalize within the problem domain. (From a statistical viewpoint, outside the problem domain it increases bias while within the problem domain it decreases variance). The resulting distance measures are similar to some of those hypothesized for cognition.

The distance measures and learning problem (clustering) are formulated as objective functions. Fast minimization of these objectives is achieved by a combination of optimization techniques - algebraic transformations, iterative projective scaling, clocked objectives, and deterministic annealing. Combining these techniques significantly increases the size of problems which may be solved with recurrent network architectures [Rangarajan et al., 1994]. Even on single-cpu workstations non-linear objectives with a million variables can routinely be minimized. With these methods we learn prototypical examples of 2-D points set and graphs from randomly generated experimental data.

## 2  Distance Measures in Unsupervised Learning

### 2.1  An Affine Invariant Point Matching Distance Measure

The first distance measure quantifies the degree of dissimilarity between two unlabeled 2-D point images, irrespective of bounded affine transformations, i.e. differences in position, orientation, scale and shear. The two images may have different numbers of points. The measure is calculated with an objective that can be used to find correspondence and pose for unlabeled feature matching in vision. Given two sets of points $\{X_j\}$ and $\{Y_k\}$, one can minimize the following objective to find the affine transformation and permutation which best maps $Y$ onto $X$ :

$$E_{pm}(m, t, A) = \sum_{j=1}^{J}\sum_{k=1}^{K} m_{jk}\|X_j - t - A \cdot Y_k\|^2 + g(A) - \alpha\sum_{j=1}^{J}\sum_{k=1}^{K} m_{jk}$$

with constraints: $\forall j \ \sum_{k=1}^{K} m_{jk} \leq 1$ , $\forall k \ \sum_{j=1}^{J} m_{jk} \leq 1$ , $\forall jk \ m_{jk} \geq 0$.

$A$ is decomposed into scale, rotation, vertical shear and oblique shear components. $g(A)$ regularizes our affine transformation - bounding the scale and shear components. $m$ is a fuzzy correspondence matrix which matches points in one image with corresponding points in the other image. The inequality constraint on $m$ allows for null matches - that is a given point in one image may match to no corresponding point in the other image. The $\alpha$ term biases the objective towards matches.

Then given two sets of points $\{X_j\}$ and $\{Y_k\}$ the distance between them is defined as:

$$D(\{X_j\}, \{Y_k\}) = \min_{m,t,A} (E_{pm}(m,t,A) \mid \text{constraints on } m)$$

This measure is an example of a more general image distance measure derived in [Mjolsness, 1992]:

$$d(x,y) = \min_T d(x, T(y)) \in [0, \infty)$$

where $T$ is a set of transformation parameters introduced by a visual grammar.

Using slack variables, and following the treatment in [Peterson and Söderberg, 1989; Yuille and Kosowsky, 1994] we employ Lagrange multipliers and an $x \log x$ barrier function to enforce the constraints with the following objective:

$$E_{pm}(m,t,A) = \sum_{j=1}^{J} \sum_{k=1}^{K} m_{jk} \|X_j - t - A \cdot Y_k\|^2 + g(A) - \alpha \sum_{j=1}^{J} \sum_{k=1}^{K} m_{jk}$$

$$+ \frac{1}{\beta} \sum_{j=1}^{J+1} \sum_{k=1}^{K+1} m_{jk}(\log m_{jk} - 1) + \sum_{j=1}^{J} \mu_j (\sum_{k=1}^{K+1} m_{jk} - 1) + \sum_{k=1}^{K} \nu_k (\sum_{j=1}^{J+1} m_{jk} - 1) \quad (1)$$

In this objective we are looking for a saddle point. (1) is minimized with respect to $m$, $t$, and $A$ which are the correspondence matrix, translation, and affine transform, and is maximized with respect to $\mu$ and $\nu$, the Lagrange multipliers that enforce the row and column constraints for $m$.

The above can be used to define many different distance measures, since given the decomposition of $A$ it is trivial to construct measures which are invariant only under some subset of the transformations (such as rotation and translation). The regularization and $\alpha$ terms may also be individually adjusted in an appropriate fashion for a specific problem domain.

## 2.2 Weighted Graph Matching Distance Measures

The following distance measure quantifies the degree of dissimilarity between two unlabeled weighted graphs. Given two graphs, represented by adjacency matrices $G_{ab}$ and $g_{ij}$, one can minimize the objective below to find the permutation which best maps $G$ onto $g$:

$$E_{gm}(m) = \sum_{a=1}^{A} \sum_{i=1}^{I} (\sum_{b=1}^{B} G_{ab} m_{bi} - \sum_{j=1}^{J} m_{aj} g_{ji})^2$$

with constraints: $\forall a \sum_{i=1}^{I} m_{ai} = 1$ , $\forall i \sum_{a=1}^{A} m_{ai} = 1$ , $\forall ai \ m_{ai} \geq 0$. These constraints are enforced in the same fashion as in (1). An algebraic fixed-point transformation and self-amplification term further transform the objective to:

$$E_{gm}(m) = \sum_{a=1}^{A} \sum_{i=1}^{I} (\mu_{ai}(\sum_{b=1}^{B} G_{ab}m_{bi} - \sum_{j=1}^{J} m_{aj}g_{ji}) - \frac{1}{2}\mu_{ai}^2 - \gamma\sigma_{ai}m_{ai} + \frac{\gamma}{2}\sigma_{ai}^2)$$

$$+ \frac{1}{\beta}\sum_{a=1}^{A} \sum_{i=1}^{I} m_{ai}(\log m_{ai} - 1) + \sum_{a=1}^{A} \kappa_a(\sum_{i=1}^{I} m_{ai} - 1) + \sum_{i=1}^{I} \lambda_i(\sum_{a=1}^{A} m_{ai} - 1) \quad (2)$$

In this objective we are also looking for a saddle point.

A second, functionally equivalent, graph matching objective is also used in the clustering problem:

$$E_{gm'}(m) = \sum_{a=1}^{A} \sum_{b=1}^{B} \sum_{i=1}^{I} \sum_{j=1}^{J} m_{ai}m_{bj}(G_{ab} - g_{ji})^2 \quad (3)$$

with constraints: $\forall a \sum_{i=1}^{I} m_{ai} = 1$ , $\forall i \sum_{a=1}^{A} m_{ai} = 1$ , $\forall ai \ m_{ai} \geq 0$.

## 2.3  The Clustering Objective

The learning problem is formulated as follows: Given a set of $I$ objects, $\{X_i\}$ find a set of $A$ cluster centers $\{Y_a\}$ and match variables $\{M_{ia}\}$ defined as

$$M_{ia} = \begin{cases} 1 & \text{if } X_i \text{ is in } Y_a\text{'s cluster} \\ 0 & \text{otherwise,} \end{cases}$$

such that each object is in only one cluster, and the total distance of all the objects from their respective cluster centers is minimized. To find $\{Y_a\}$ and $\{M_{ia}\}$ minimize the cost function,

$$E_{cluster}(Y, M) = \sum_{i=1}^{I} \sum_{a=1}^{A} M_{ia}D(X_i, Y_a)$$

with the constraint that $\forall i \sum_a M_{ia} = 1$ , $\forall ai \ M_{ai} \geq 0$. $D(X_i, Y_a)$, the distance function, is a measure of dissimilarity between two objects.

The constraints on $M$ are enforced in a manner similar to that described for the distance measure, except that now only the rows of the matrix $M$ need to add to one, instead of both the rows and the columns.

$$E_{cluster}(Y, M) = \sum_{i=1}^{I} \sum_{a=1}^{A} M_{ia}D(X_i, Y_a) + \frac{1}{\beta}\sum_{i=1}^{I} \sum_{a=1}^{A} M_{ia}(\log M_{ia} - 1)$$

$$+ \sum_{i=1}^{I} \lambda_i(\sum_{a=1}^{A} M_{ia} - 1) \quad (4)$$

Here, the objects are point-sets or weighted graphs. If point-sets the distance measure $D(X_i, Y_a)$ is replaced by (1), if graphs it is replaced by (2) or (3).

Therefore, given a set of objects, $X$, we construct $E_{cluster}$ and upon finding the appropriate saddle point of that objective, we will have $Y$, their cluster centers, and $M$, their cluster memberships.

## 3  The Algorithm

The algorithm to minimize the clustering objectives consists of two loops - an inner loop to minimize the distance measure objective [either (1) or (2)] and an outer loop to minimize the clustering objective (4). Using coordinate descent in the outer loop results in dynamics similar to the EM algorithm [Jordan and Jacobs, 1994] for clustering. All variables occurring in the distance measure objective are held fixed during this phase. The inner loop uses coordinate ascent/descent which results in repeated row and column projections for $m$. The minimization of $m$, and the distance measure variables [either $t$, $A$ of (1) or $\mu$, $\sigma$ of (2)], occurs in an incremental fashion, that is their values are saved after each inner loop call from within the outer loop and are then used as initial values for the next call to the inner loop. This tracking of the values of the distance measure variables in the inner loop is essential to the efficiency of the algorithm since it greatly speeds up each inner loop optimization. Most coordinate ascent/descent phases are computed analytically, further speeding up the algorithm. Some local minima are avoided, by deterministic annealing in both the outer and inner loops. The multi-phase dynamics maybe described as a clocked objective. Let $\{D\}$ be the set of distance measure variables excluding $m$. The algorithm is as follows:

Initialize $\{D\}$ to the equivalent of an identity transform, $Y$ to random values
**Begin** Outer Loop
  **Begin** Inner Loop
    Initialize $\{D\}$ with previous values
    Find $m$, $\{D\}$ for each $ia$ pair :
      Find $m$ by softmax, projecting across $j$, then $k$, iteratively
      Find $\{D\}$ by coordinate descent
  **End** Inner Loop
  Find $M$,$Y$ using fixed values of $m$, $\{D\}$, determined in inner loop:
    Find $M$ by softmax, across $i$
    Find $Y$ by coordinate descent
  Increase $\beta_M$, $\beta_m$
**End** Outer Loop

When analytic solutions are computed for $Y$ the outer loop takes a form similar to fuzzy ISODATA clustering, with annealing on the fuzziness parameter.

## 4  Methods and Experimental Results

Four series of experiments were ran with randomly generated data to evaluate the learning algorithms. Point sets were clustered in the first three experiments and weighted graphs were clustered in the fourth. In each experiment a set of object

models were randomly generated. Then from each object model a set of object instances were created by transforming the object model according to the problem domain assumed for that experiment. For example, an object represented by points in two dimensional space was translated, rotated, scaled, sheared, and permuted to form a new point set. A object represented by a weighted graph was permuted. Noise was added to further distort the object. Parts of the object were deleted and spurious features (points) were added. In this manner, from a set of object models, a larger number of object instances were created. Then with no knowledge of the original objects models or cluster memberships, we clustered the object instances using the algorithms described above.

The results were evaluated by comparing the object prototypes (cluster centers) formed by each experimental run to the object models used to generate the object instances for that experiment. The distance measures used in the clustering were used for this comparison, i.e. to calculate the distance between the learned prototype and the original object. Note that this distance measure also incorporates the transformations used to create the object instances. The mean and standard deviations of these distances were plotted (Figure 1) over hundreds of experiments, varying the object instance generation noise. The straight line appearing on each graph displays the effect of the noise only. It is the expected object model-object prototype distance if no transformations were applied, no features were deleted or added, and the cluster memberships of the object instances were known. It serves as an absolute lower bound on our learning algorithm. The noise was increased in each series of experiments until the curve flattened - that is the object instances became so distorted by noise that no information about the original objects could be recovered by the algorithm.

In the first series of experiments (Figure 1a), point set objects were translated, rotated, scaled, and permuted. Initial object models were created by selecting points with a uniform distribution within a unit square. The transformations to create the object instance were selected with a uniform distribution within the following bounds; translation: $\pm.5$, rotation: $\pm 27°$, log($scale$): $\pm \log(.5)$. 100 object instances were generated from 10 object models. All objects contained 20 points. The standard deviation of the Gaussian noise was varied by .02 from .02 to .16. 15 experiments were run at each noise level. The data point at each error bar represents 150 distances (15 experiments times 10 model-prototype distances for each experiment).

In the second and third series of experiments (Figures 1b and 1c), point set objects were translated, rotated, scaled, sheared (obliquely and vertically), and permuted. Each object point had a 10% probability of being deleted and a 5% probability of generating a spurious point. The point sets and transformations were randomly generated as in the first experiment, except for these bounds; log($scale$): $\pm \log(.7)$, log($verticalshear$): $\pm \log(.7)$, and log($obliqueshear$): $\pm \log(.7)$. In experiment 2, 64 object instances and 4 object models of 15 points each were used. In experiment 3, 256 object instances and 8 object models of 20 points each were used. Noise levels like experiment 1 were used, with 20 experiments run at each noise level in experiment 2 and 10 experiments run at each noise level in experiment 3.

In experiment 4 (Figure 1d), object models were represented by fully connected weighted graphs. The link weights in the initial object models were selected with a uniform distribution between 0 and 1. The objects were then randomly permuted

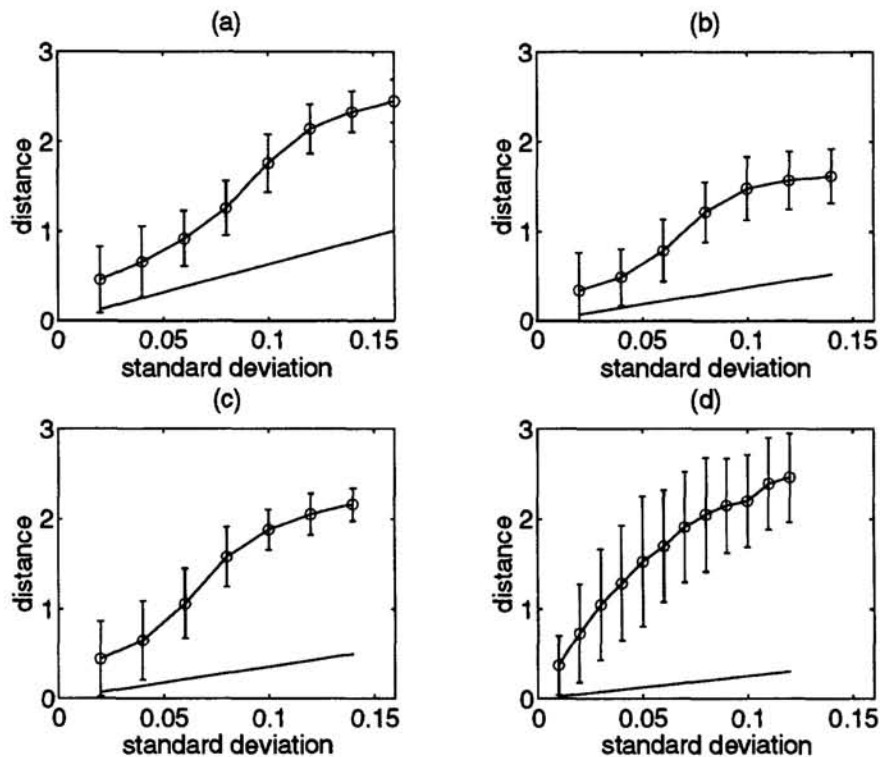

Figure 1: (a): 10 clusters, 100 point sets, 20 points each, scale ,rotation, translation, 120 experiments (b): 4 clusters, 64 point sets, 15 points each, affine, 10 % deleted, 5 % spurious, 140 experiments (c): 8 clusters, 256 point sets, 20 points each, affine, 10 % deleted, 5 % spurious, 70 experiments (d): 4 clusters, 64 graphs, 10 nodes each, 360 experiments

to form the object instance and uniform noise was added to the link weights. 64 object instances were generated from 4 object models consisting of 10 node graphs with 100 links. The standard deviation of the noise was varied by .01 from .01 to .12. 30 experiments where run at each noise level.

In most experiments at low noise levels ($\leq$ .06 for point sets, $\leq$ .03 for graphs), the object prototypes learned were very similar to the object models. Even at higher noise levels object prototypes similar to the object models are formed, though less consistently. Results from about 700 experiments are plotted. The objective for experiment 3 contained close to one million variables and converged in about 4 hours on an SGI Indigo workstation. The convergence times of the objectives of experiments 1, 2 and 4 were 120, 10 and 10 minutes respectively.

## 5 Conclusions

It has long been argued by many, that learning in complex domains typically associated with human intelligence requires some type of prior structure or knowledge. We have begun to develop a set of tools that will allow the incorporation of prior

structure within learning. Our models incorporate many features needed in complex domains like vision - noise, missing and spurious features, non-rigid transformations. They can learn objects with inherent structure, like graphs. Many experiments have been run on experimentally generated data sets. Several directions for future research hold promise. One might be the learning of OCR data [Gold et al., 1995]. Second a supervised learning stage could be added to our algorithms. Finally the power of the distance measures can be enhanced to operate on attributed relational graphs with deleted nodes and links [Gold and Rangarajan, 1995].

## Acknowledgements

ONR/DARPA: N00014-92-J-4048, AFOSR: F49620-92-J-0465 and Yale CTAN.

## Footnotes

[1] E-mail address of authors: lastname-firstname@cs.yale.edu

[2] Department of Computer Science and Engineering, University of California at San Diego (UCSD), La Jolla, CA 92093-0114. E-mail: emj@cs.ucsd.edu

## References

S. Geman, E. Bienenstock, and R. Doursat. (1992) Neural networks and the bias/variance dilemma. *Neural Computation* 4:1-58.

S. Gold, E. Mjolsness and A. Rangarajan. (1994) Clustering with a domain-specific distance measure. In J. Cowan *et al.*, (eds.), *NIPS 6*. Morgan Kaufmann.

S. Gold, C. P. Lu, A. Rangarajan, S. Pappu and E. Mjolsness. (1995) New algorithms for 2D and 3D point matching: pose estimation and correspondence. In G. Tesauro *et al.*, (eds.), *NIPS 7*. San Francisco, CA: Morgan Kaufmann.

S. Gold and A. Rangarajan (1995) A graduated assignment algorithm for graph matching. YALEU/DCS/TR-1062, Yale Univ., CS Dept.

M. I. Jordan and R. A. Jacobs. (1994) Hierarchical mixtures of experts and the EM algorithm. *Neural Computation*, 6:181-214.

E. Mjolsness. Visual grammars and their neural networks. (1992) *SPIE Conference on the Science of Artificial Neural Networks*, **1710**:63-85.

C. Peterson and B. Söderberg. A new method for mapping optimization problems onto neural networks. (1989) *International Journal of Neural Systems*,1(1):3-22.

A. Rangarajan, S. Gold and E. Mjolsness. (1994) A novel optimizing network architecture with applications. YALEU/DCS/TR-1036, Yale Univ., CS Dept.

R. Shepard. (1989). Internal representation of universal regularities: A challenge for connectionism. In L. Nadel *et al.*, (eds.), *Neural Connections, Mental Computation*. Cambridge, MA, London, England: Bradford/MIT Press.

P. Simard, Y. Le Cun, and J. Denker. (1993) Efficient pattern recognition using a transformation distance. In S. Hanson *et al.*, (eds.), *NIPS 5*. San Mateo, CA: Morgan Kaufmann.

C. Williams, R. Zemel, and M. Mozer. (1993) Unsupervised learning of object models. AAAI Tech. Rep. FSS-93-04, Univ. of Toronto, CS Dept.

A. L. Yuille and J.J. Kosowsky. (1994). Statistical physics algorithms that converge. *Neural Computation*, 6:341-356.